# Composite Multiclass Losses

**Elodie Vernet**
ENS Cachan
evernet@ens-cachan.fr

**Robert C. Williamson**
ANU and NICTA
Bob.Williamson@anu.edu.au

**Mark D. Reid**
ANU and NICTA
Mark.Reid@anu.edu.au

## Abstract

We consider loss functions for multiclass prediction problems. We show when a multiclass loss can be expressed as a "proper composite loss", which is the composition of a proper loss and a link function. We extend existing results for binary losses to multiclass losses. We determine the stationarity condition, Bregman representation, order-sensitivity, existence and uniqueness of the composite representation for multiclass losses. We subsume existing results on "classification calibration" by relating it to properness and show that the simple integral representation for binary proper losses can not be extended to multiclass losses.

## 1 Introduction

The motivation of this paper is to understand the intrinsic structure and properties of suitable loss functions for the problem of multiclass prediction, which includes *multiclass probability estimation*. Suppose we are given a data sample $S := (x_i, y_i)_{i \in [m]}$ where $x_i \in \mathscr{X}$ is an observation and $y_i \in \{1, .., n\} =: [n]$ is its corresponding class. We assume the sample $S$ is drawn iid according to some distribution $\mathbb{P} = \mathbb{P}_{\mathscr{X}, \mathscr{Y}}$ on $\mathscr{X} \times [n]$. Given a new observation $x$ we want to predict the probability $p_i := \mathbb{P}(Y = i | X = x)$ of $x$ belonging to class $i$, for $i \in [n]$. *Multiclass classification* requires the learner to predict the most likely class of $x$; that is to find $\hat{y} = \arg\max_{i \in [n]} p_i$.

A loss measures the quality of prediction. Let $\Delta^n := \{(p_1, \ldots, p_n) : \sum_{i \in [n]} p_i = 1, \text{and } 0 \le p_i \le 1, \forall i \in [n]\}$ denote the *n-simplex*. For multiclass probability estimation, $\ell : \Delta^n \to \mathbb{R}^n_+$. For classification, the loss $\ell : [n] \to \mathbb{R}^n_+$. The *partial losses* $\ell_i$ are the components of $\ell(q) = (\ell_1(q), \ldots, \ell_n(q))'$.

Proper losses are particularly suitable for probability estimation. They have been studied in detail when $n = 2$ (the "binary case") where there is a nice integral representation [1, 2, 3], and characterization [4] when differentiable. Classification calibrated losses are an analog of proper losses for the problem of classification [5]. The relationship between classification calibration and properness was determined in [4] for $n = 2$. Most of these results have had no multiclass analogue until now.

The design of losses for multiclass prediction has received recent attention [6, 7, 8, 9, 10, 11, 12] although none of these papers developed the connection to proper losses, and most restrict consideration to margin losses (which imply certain symmetry conditions). Glasmachers [13] has shown that certain learning algorithms can still behave well when the losses do not satisfy the conditions in these earlier papers because the requirements are actually stronger than needed.

Our contributions are: We relate properness, classification calibration, and the notion used in [8] which we rename "prediction calibrated" §3; we provide a novel characterization of multiclass properness §4; we study composite proper losses (the composition of a proper loss with an invertible link) presenting new uniqueness and existence results §5; we show how the above results can aid in the design of proper losses §6; and we present a (somewhat surprising) negative result concerning the integral representation of proper multiclass losses §7. Many of our results are characterisations. Full proofs are provided in the extended version [14].

## 2   Formal Setup

Suppose $\mathscr{X}$ is some set and $\mathscr{Y} = \{1,\ldots,n\} = [n]$ is a set of labels. We suppose we are given data $(x_i, y_i)_{i \in [m]}$ such that $Y_i \in \mathscr{Y}$ is the label corresponding to $x_i \in \mathscr{X}$. These data follow a joint distribution $\mathbb{P}_{\mathscr{X},\mathscr{Y}}$. We denote by $\mathbb{E}_{\mathscr{X},\mathscr{Y}}$ and $\mathbb{E}_{\mathscr{Y}|\mathscr{X}}$ respectively, the expectation and the conditional expectation with respect to $\mathbb{P}_{\mathscr{X},\mathscr{Y}}$.

The *conditional risk L* associated with a loss $\ell$ is the function

$$L \colon \Delta^n \times \Delta^n \ni (p,q) \mapsto L(p,q) = \mathbb{E}_{\mathsf{Y} \sim p}\ell_{\mathsf{Y}}(q) = p' \cdot \ell(q) = \sum_{i \in [n]} p_i \ell_i(q) \in \mathbb{R}_+,$$

where $\mathsf{Y} \sim p$ means $\mathsf{Y}$ is drawn according to a multinomial distribution with parameter $p$. In a typical learning problem one will make an estimate $q \colon \mathscr{X} \to \Delta^n$. The *full risk* is $\mathbb{L}(q) = \mathbb{E}_{\mathscr{X}}\mathbb{E}_{\mathscr{Y}|\mathscr{X}}\ell_{\mathsf{Y}}(q(\mathsf{X}))$.

Minimizing $\mathbb{L}(q)$ over $q \colon \mathscr{X} \to \Delta^n$ is equivalent to minimizing $L(p(x), q(x))$ over $q(x) \in \Delta^n$ for all $x \in \mathscr{X}$ where $p(x) = (p_1(x), \ldots, p_n(x))'$, $p'$ is the transpose of $p$, and $p_i(x) = \mathbb{P}(\mathsf{Y} = i | \mathsf{X} = x)$. Thus it suffices to only consider the conditional risk; confer [3].

A loss $\ell \colon \Delta^n \to \mathbb{R}^n_+$ is *proper* if $L(p,p) \leq L(p,q)$, $\forall p,q \in \Delta^n$. It is *strictly proper* if the inequality is strict when $p \neq q$. The *conditional Bayes risk* $\underline{L} \colon \Delta^n \ni p \mapsto \inf_{q \in \Delta^n} L(p,q)$. This function is always concave [2]. If $\ell$ is proper, then $\underline{L}(p) = L(p,p) = p' \cdot \ell(p)$. Strictly proper losses induce *Fisher consistent* estimators of probabilities: if $\ell$ is strictly proper, $p = \arg\min_q L(p,q)$.

In order to differentiate the losses we project the $n$-simplex into a subset of $\mathbb{R}^{n-1}$. We denote by $\Pi_\Delta \colon \Delta^n \ni p = (p_1, \ldots, p_n)' \mapsto \tilde{p} = (p_1, \ldots, p_{n-1})' \in \tilde{\Delta}^n := \{(p_1, \ldots, p_{n-1})' \colon p_i \geq 0, \forall i \in [n], \sum_{i=1}^{n-1} p_i \leq 1\}$, the projection of the $n$-simplex $\Delta^n$, and $\Pi_\Delta^{-1} \colon \tilde{\Delta}^n \ni \tilde{p} = (\tilde{p}_1, \ldots, \tilde{p}_{n-1}) \mapsto p = (\tilde{p}_1, \ldots, \tilde{p}_{n-1}, 1 - \sum_{i=1}^{n-1} \tilde{p}_i)' \in \Delta^n$ its inverse.

The losses above are defined on the simplex $\Delta^n$ since the argument (an estimator) represents a probability vector. However it is sometimes desirable to use another set $\mathscr{V}$ of predictions. One can consider losses $\ell \colon \mathscr{V} \to \mathbb{R}^n_+$. Suppose there exists an invertible function $\psi \colon \Delta^n \to \mathscr{V}$. Then $\ell$ can be written as a composition of a loss $\lambda$ defined on the simplex with $\psi^{-1}$. That is, $\ell(v) = \lambda^\psi(v) := \lambda(\psi^{-1}(v))$. Such a function $\lambda^\psi$ is a *composite loss*. If $\lambda$ is proper, we say $\ell$ is a *proper composite loss*, with *associated proper loss* $\lambda$ and *link* $\psi$.

We use the following notation. The $k$th unit vector $e_k$ is the $n$ vector with all components zero except the $k$th which is 1. The $n$-vector $\mathbb{1}_n := (1, \ldots, 1)'$. The derivative of a function $f$ is denoted $\mathsf{D}f$ and its Hessian $\mathsf{H}f$. Let $\mathring{\Delta}^n := \{(p_1, \ldots, p_n) \colon \sum_{i \in [n]} p_i = 1, \text{and } 0 < p_i < 1, \forall i \in [n]\}$ and $\partial \Delta^n := \Delta^n \setminus \mathring{\Delta}^n$.

## 3   Relating Properness to Classification Calibration

Properness is an attractive property of a loss for the task of class probability estimation. However if one is merely interested in *classifying* (predicting $\hat{y} \in [n]$ given $x \in \mathscr{X}$) then one requires less. We relate *classification calibration* (the analog of properness for classification problems) to properness.

Suppose $c \in \mathring{\Delta}^n$. We cover $\Delta^n$ with $n$ subsets each representing one class:

$$\mathscr{T}_i(c) := \{p \in \Delta^n \colon \forall j \neq i \; p_i c_j \geq p_j c_i\}.$$

Observe that for $i \neq j$, the sets $\{p \in \mathbb{R} \colon p_i c_j = p_j c_i\}$ are subsets of dimension $n-2$ through $c$ and all $e_k$ such that $k \neq i$ and $k \neq j$. These subsets partition $\Delta^n$ into two parts, the subspace $\mathscr{T}_i$ is the intersection of the subspaces delimited by the precedent $(n-2)$-subspace and in the same side as $e_i$. We will make use of the following properties of $\mathscr{T}_i(c)$.

**Lemma 1** *Suppose $c \in \mathring{\Delta}^n$, $i \in [n]$. Then the following hold:*

1. *For all $p \in \Delta^n$, there exists $i$ such that $p \in \mathscr{T}_i(c)$.*

2. *Suppose $p \in \Delta^n$. $\mathscr{T}_i(c) \cap \mathscr{T}_j(c) \subseteq \{p \in \Delta^n \colon p_i c_j = p_j c_i\}$, a subspace of dimension $n-2$.*

3. *Suppose $p \in \Delta^n$. If $p \in \bigcap_{i=1}^n \mathscr{T}_i(c)$ then $p = c$.*

4. *For all $p, q \in \Delta^n$, $p \neq q$, there exists $c \in \mathring{\Delta}^n$, and $i \in [n]$ such that $p \in \mathscr{T}_i(c)$ and $q \notin \mathscr{T}_i(c)$.*

Classification calibrated losses have been developed and studied under some different definitions and names [6, 5]. Below we generalise the notion of $c$-calibration which was proposed for $n = 2$ in [4] as a generalisation of the notion of classification calibration in [5].

**Definition 2** *Suppose $\ell\colon \Delta^n \to \mathbb{R}_+^n$ is a loss and $c \in \mathring{\Delta}^n$. We say $\ell$ is $c$-calibrated at $p \in \Delta^n$ if for all $i \in [n]$ such that $p \notin \mathscr{T}_i(c)$ then $\forall q \in \mathscr{T}_i(c)$, $\underline{L}(p) < L(p,q)$. We say that $\ell$ is $c$-calibrated if $\forall p \in \Delta^n$, $\ell$ is $c$-calibrated at $p$.*

Definition 2 means that if the probability vector $q$ one predicts doesn't belong to the same subset (i.e. doesn't predict the same class) as the real probability vector $p$, then the loss might be larger.

Classification calibration in the sense used in [5] corresponds to $\frac{1}{2}$-calibrated losses when $n = 2$. If $c_{\text{mid}} := (\frac{1}{n}, \ldots, \frac{1}{n})'$, $c_{\text{mid}}$-calibration induces Fisher-consistent estimates in the case of classification. Furthermore "$\ell$ is $c_{\text{mid}}$-calibrated and for all $i \in [n]$, and $\ell_i$ is continuous and bounded below" is equivalent to "$\ell$ is infinite sample consistent as defined by [6]". This is because if $\ell$ is continuous and $\mathscr{T}_i(c)$ is closed, then $\forall q \in \mathscr{T}_i(c)$, $\underline{L}(p) < L(p,q)$ if and only if $\underline{L}(p) < \inf_{q \in \mathscr{T}_i(c)} L(p,q)$.

The following result generalises the correspondence between binary classification calibration and properness [4, Theorem 16] to multiclass losses ($n > 2$).

**Proposition 3** *A continuous loss $\ell\colon \Delta^n \to \mathbb{R}_+^n$ is strictly proper if and only if it is $c$-calibrated for all $c \in \mathring{\Delta}^n$.*

In particular, a continuous strictly proper loss is $c_{\text{mid}}$-calibrated. Thus for any estimator $\hat{q}_n$ of the conditional probability vector one constructs by minimizing the empirical average of a continuous strictly proper loss, one can build an estimator of the label (corresponding to the largest probability of $\hat{q}_n$) which is Fisher consistent for the problem of classification.

In the binary case, $\ell$ is classification calibrated if and only if the following implication holds [5]:

$$\left( \mathbb{L}(f_n) \to \min_g \mathbb{L}(g) \right) \Rightarrow \left( \mathbb{P}_{\mathscr{X},\mathscr{Y}}(\mathsf{Y} \neq f_n(\mathsf{X})) \to \min_g \mathbb{P}_{\mathscr{X},\mathscr{Y}}(\mathsf{Y} \neq g(\mathsf{X})) \right). \tag{1}$$

Tewari and Bartlett [8] have characterised when (1) holds in the multiclass case. Since there is no reason to assume the equivalence between classification calibration and (1) still holds for $n > 2$, we give different names for these two notions. We keep the name of classification calibration for the notion linked to Fisher consistency (as defined before) and call prediction calibrated the notion of Tewari and Bartlett (equivalent to (1)).

**Definition 4** *Suppose $\ell\colon \mathscr{V} \to \mathbb{R}_+^n$ is a loss. Let $\mathscr{C}_\ell = \mathrm{co}(\{\ell(v)\colon v \in \mathscr{V}\})$, the convex hull of the image of $\mathscr{V}$. $\ell$ is said to be* prediction calibrated *if there exists a prediction function $\mathrm{pred}\colon \mathbb{R}^n \to [n]$ such that*

$$\forall p \in \Delta^n\colon \inf_{z \in \mathscr{C}_\ell, p_{\mathrm{pred}(z)} < \max_i p_i} p' \cdot z > \inf_{z \in \mathscr{C}_\ell} p' \cdot z = \underline{L}(p).$$

Observe that the class is predicted from $\ell(p)$ and not directly from $p$ (which is equivalent if the loss is invertible). Suppose that $\ell\colon \Delta^n \to \mathbb{R}_+^n$ is such that $\ell$ is prediction calibrated and $\mathrm{pred}(\ell(p)) \in \arg\max_i p_i$. Then $\ell$ is $c_{\text{mid}}$-calibrated almost everywhere.

By introducing a reference "link" $\bar{\psi}$ (which corresponds to the actual link if $\ell$ is a proper composite loss) we now show how the pred function can be canonically expressed in terms of $\arg\max_i p_i$.

**Proposition 5** *Suppose $\ell\colon \mathscr{V} \to \mathbb{R}_+^n$ is a loss. Let $\bar{\psi}(p) \in \arg\min_{v \in \mathscr{V}} L(p,v)$ and $\lambda = \ell \circ \bar{\psi}$. Then $\lambda$ is proper. If $\ell$ is prediction calibrated then $\mathrm{pred}(\lambda(p)) \in \arg\max_i p_i$.*

## 4 Characterizing Properness

We first present some simple (but new) consequences of properness. We say $f\colon C \subset \mathbb{R}^n \to \mathbb{R}^n$ is *monotone* on $C$ when for all $x$ and $y$ in $C$, $(f(x) - f(y))' \cdot (x - y) \geq 0$; confer [15].

**Proposition 6** *Suppose $\ell\colon \Delta^n \to \mathbb{R}_+^n$ is a loss. If $\ell$ is proper, then $-\ell$ is monotone.*

**Proposition 7** *If $\ell$ is strictly proper then it is invertible.*

A theme of the present paper is the extensibility of results concerning binary losses to multiclass losses. The following proposition shows how the characterisation of properness in the general (not necessarily differentiable) multiclass case can be reduced to the binary case. In the binary case, the two classes are often denoted $-1$ and $1$ and the loss is denoted $\ell = (\ell_1, \ell_{-1})'$. We project the 2-simplex $\Delta^2$ into $[0,1]$: $\eta \in [0,1]$ is the projection of $(\eta, 1-\eta) \in \Delta^2$.

**Proposition 8** *Suppose $\ell \colon \Delta^n \to \mathbb{R}_+^n$ is a loss. Define*

$$\tilde{\ell}^{p,q} \colon [0,1] \ni \eta \mapsto \left( \begin{array}{c} \tilde{\ell}_1^{p,q}(\eta) \\ \tilde{\ell}_{-1}^{p,q}(\eta) \end{array} \right) = \left( \begin{array}{c} q' \cdot \ell\big(p + \eta(q-p)\big) \\ p' \cdot \ell\big(p + \eta(q-p)\big) \end{array} \right).$$

*Then $\ell$ is (strictly) proper if and only if $\tilde{\ell}^{p,q}$ is (strictly) proper $\forall p,q \in \partial\Delta^n$.*

This proposition shows that in order to check if a loss is proper one needs only to check the properness in each line. One could use the easy characterization of properness for differentiable binary losses ($\ell \colon [0,1] \to \mathbb{R}_+^2$ is proper if and only if $\forall \eta \in [0,1]$, $\frac{-\ell_1'(\eta)}{1-\eta} = \frac{\ell_{-1}'(\eta)}{\eta} \geq 0$, [4]). However this needs to be checked for all lines defined by $p,q \in \partial\Delta^n$. We now extend some characterisations of properness to the multiclass case by using Proposition 8.

Lambert [16] proved that in the binary case, properness is equivalent to the fact that the further your prediction is from reality, the larger the loss ("order sensitivity"). The result relied upon on the total order of $\mathbb{R}$. In the multiclass case, there does not exist such a total order. Yet, one can compare two predictions if they are in the same line as the true real class probability. The next result is a generalization of the binary case equivalence of properness and order sensitivity.

**Proposition 9** *Suppose $\ell \colon \Delta^n \to \mathbb{R}_+^n$ is a loss. Then $\ell$ is (strictly) proper if and only if $\forall p,q \in \Delta^n$, $\forall 0 \leq h_1 \leq h_2$, $L(p, p + h_1(q-p)) \leq L(p, p + h_2(q-p))$ (the inequality is strict if $h_1 \neq h_2$).*

"Order sensitivity" tells us more about properness: the true class probability minimizes the risk and if the prediction moves away from the true class probability in a line then the risk increases. This property appears convenient for optimization purposes: if one reaches a local minimum in the second argument of the risk and the loss is strictly proper then it is a global minimum. If the loss is proper, such a local minimum is a global minimum or a constant in an open set. But observe that typically one is minimising the full risk $\mathbb{L}(q(\cdot))$ over functions $q \colon \mathscr{X} \to \Delta^n$. Order sensitivity of $\ell$ does *not* imply this optimisation problem is well behaved; one needs convexity of $q \mapsto L(p,q)$ for all $p \in \Delta^n$ to ensure convexity of the functional optimisation problem.

The order sensitivity along a line leads to a new characterisation of differentiable proper losses. As in the binary case, one condition comes from the fact that the derivative is zero at a minimum and the other ensures that it is really a minimum.

**Corollary 10** *Suppose $\ell \colon \Delta^n \to \mathbb{R}_+^n$ is a loss such that $\tilde{\ell} = \ell \circ \Pi_\Delta^{-1}$ is differentiable. Let $M(p) = \mathrm{D}\tilde{\ell}(\Pi_\Delta(p)) \cdot \mathrm{D}\Pi_\Delta(p)$. Then $\ell$ is proper if and only if*

$$\left. \begin{array}{rcl} p' \cdot M(p) & = & 0 \\ (q-r)' \cdot M(p) \cdot (q-r) & \leq & 0 \end{array} \right\} \ \forall q,r \in \Delta^n, \ \forall p \in \mathring{\Delta}^n. \tag{2}$$

We know that for any loss, its Bayes risk $\underline{L}(p) = \inf_{q \in \Delta^n} L(p,q) = \inf_{q \in \Delta^n} p' \cdot \ell(q)$ is concave. If $\ell$ is proper, $\underline{L}(p) = p' \cdot \ell(p)$. Rather than working with the loss $\ell \colon \mathscr{V} \to \mathbb{R}_+^n$ we will now work with the simpler associated conditional Bayes risk $\underline{L} \colon \mathscr{V} \to \mathbb{R}_+$.

We need two definitions from [15]. Suppose $f \colon \mathbb{R}^n \to \mathbb{R}$ is concave. Then $\lim_{t\downarrow 0} \frac{f(x+td)-f(x)}{t}$ exists, and is called the *directional derivative* of $f$ at $x$ in the direction $d$ and is denoted $\mathrm{D}f(x,d)$. By analogy with the usual definition of *sub*differential, the *superdifferential* $\partial f(x)$ of $f$ at $x$ is

$$\partial f(x) := \big\{ s \in \mathbb{R}^n \colon s' \cdot y \geq \mathrm{D}f(x,y), \ \forall y \in \mathbb{R}^n \big\} = \big\{ s \in \mathbb{R}^n \colon f(y) \leq f(x) + s' \cdot (y-x), \ \forall y \in \mathbb{R}^n \big\}.$$

A vector $s \in \partial f(x)$ is called a *supergradient* of $f$ at $x$.

The next proposition is a restatement of the well known Bregman representation of proper losses; see [17] for the differentiable case, and [2, Theorem 3.2] for the general case.

**Proposition 11** *Suppose $\ell\colon \Delta^n \to \mathbb{R}_+^n$ is a loss. Then $\ell$ is proper if and only if there exists a concave function $f$ and $\forall q \in \Delta^n$, there exists a supergradient $A(q) \in \partial f(q)$ such that*

$$\forall p, q \in \Delta^n, \ p' \cdot \ell(q) = L(p,q) = f(q) + (p-q)' \cdot A(q).$$

*Then $f$ is unique and $f(p) = L(p,p) = \underline{L}(p)$.*

The fact that $f$ is defined on a simplex is not a problem. Indeed, the superdifferential becomes $\partial f(x) = \{s \in \mathbb{R}^n : s' \cdot d \geq \mathsf{D}f(x,d), \forall d \in \Delta^n\} = \{s \in \mathbb{R}^n : f(y) \leq f(x) + s' \cdot (y-x), \ \forall y \in \Delta^n\}$. If $\tilde{f} = f \circ \Pi_\Delta^{-1}$ is differentiable at $\tilde{q} \in \mathring{\Delta}^n$, $A(q) = (\mathsf{D}\tilde{f}(\Pi_\Delta(q)), 0)' + \alpha \mathbb{1}_n'$, $\alpha \in \mathbb{R}$. Then $(p-q)' \cdot A(q) = \mathsf{D}\tilde{f}(\Pi_\Delta(q)) \cdot (\Pi_\Delta(p) - \Pi_\Delta(q))$. Hence for any concave differentiable function $f$, there exists an unique proper loss whose Bayes risk is equal to $f$ (we say that $f$ is differentiable when $\tilde{f}$ is differentiable).

The last property gives us the form of the proper losses associated with a Bayes risk. Suppose $\underline{L}\colon \Delta^n \to \mathbb{R}_+$ is concave. The proper losses whose Bayes risk is equal to $\underline{L}$ are

$$\ell\colon \Delta^n \ni q \mapsto \left( \underline{L}(q) + (e_i - q)' \cdot A(q) \right)_{i=1}^n \in \mathbb{R}_+^n, \ \forall A(q) \in \partial \underline{L}(q). \tag{3}$$

This result suggests that some information is lost by representing a proper loss via its Bayes risk (when the last is not differentiable). The next proposition elucidates this by showing that proper losses which have the same Bayes risk are equal almost everywhere.

**Proposition 12** *Two proper losses $\ell^1$ and $\ell^2$ have the same conditional Bayes risk function $\underline{L}$ if and only if $\ell^1 = \ell^2$ almost everywhere. If $\underline{L}$ is differentiable, $\ell^1 = \ell^2$ everywhere.*

We say that $\underline{L}$ is differentiable at $p$ if $\tilde{\underline{L}} = \underline{L} \circ \Pi_\Delta^{-1}$ is differentiable at $\tilde{p} = \Pi_\Delta(p)$.

**Proposition 13** *Suppose $\ell\colon \Delta^n \to \mathbb{R}_+^n$ is a proper loss. Then $\ell$ is continuous in $\mathring{\Delta}^n$ if and only if $\underline{L}$ is differentiable on $\mathring{\Delta}^n$; $\ell$ is continuous at $p \in \mathring{\Delta}^n$ if and only if, $\underline{L}$ is differentiable at $p \in \mathring{\Delta}^n$.*

# 5 The Proper Composite Representation: Uniqueness and Existence

It is sometimes helpful to define a loss on some set $\mathscr{V}$ rather than $\Delta^n$; confer [4]. Composite losses (see the definition in §2) are a way of constructing such losses: given a proper loss $\lambda\colon \Delta^n \to \mathbb{R}_+^n$ and an invertible link $\psi\colon \Delta^n \to \mathscr{V}$, one defines $\lambda^\psi\colon \mathscr{V} \to \mathbb{R}_+^n$ using $\lambda^\psi = \lambda \circ \psi^{-1}$. We now consider the question: given a loss $\ell\colon \mathscr{V} \to \mathbb{R}_+^n$, when does $\ell$ have a *proper composite representation* (whereby $\ell$ can be written as $\ell = \lambda \circ \psi^{-1}$), and is this representation unique? We first consider the binary case and study the uniqueness of the representation of a loss as a proper composite loss.

**Proposition 14** *Suppose $\ell = \lambda \circ \psi^{-1}\colon \mathscr{V} \to \mathbb{R}_+^2$ is a proper composite loss and that the proper loss $\lambda$ is differentiable and the link function $\psi$ is differentiable and invertible. Then the proper loss $\lambda$ is unique. Furthermore $\psi$ is unique if $\forall v_1, v_2 \in \mathbb{R}$, $\exists v \in [v_1, v_2]$, $\ell_1'(v) \neq 0$ or $\ell_{-1}'(v) \neq 0$. If there exists $\bar{v}_1, \bar{v}_2 \in \mathbb{R}$ such that $\ell_1'(v) = \ell_{-1}'(v) = 0 \ \forall v \in [\bar{v}_1, \bar{v}_2]$, one can choose any $\psi|_{[\bar{v}_1, \bar{v}_2]}$ such that $\psi$ is differentiable, invertible and continuous in $[\bar{v}_1, \bar{v}_2]$ and obtain $\ell = \lambda \circ \psi^{-1}$, and $\psi$ is uniquely defined where $\ell$ is invertible.*

**Proposition 15** *Suppose $\ell\colon \mathscr{V} \to \mathbb{R}_+^2$ is a differentiable binary loss such that $\forall v \in \mathscr{V}$, $\ell_{-1}'(v) \neq 0$ or $\ell_1'(v) \neq 0$. Then $\ell$ can be expressed as a proper composite loss if and only if the following three conditions hold: 1) $\ell_1$ is decreasing (increasing); 2) $\ell_{-1}$ is increasing (decreasing); and 3) $f\colon \mathscr{V} \ni v \mapsto \frac{\ell_1'(v)}{\ell_{-1}'(v)}$ is strictly increasing (decreasing) and continuous.*

Observe that the last condition is alway satisfied if both $\ell_1$ and $\ell_{-1}$ are convex.

Suppose $\varphi\colon \mathbb{R} \to \mathbb{R}_+$ is a function. The loss defined via $\ell_\varphi\colon \mathscr{V} \ni v \mapsto (\ell_{-1}(v), \ell_1(v))' = (\varphi(-v), \varphi(v))' \in \mathbb{R}_+^2$ is called a binary *margin loss*. Binary margin losses are often used for classification problems. We will now show how the above proposition applies to them.

**Corollary 16** *Suppose* $\varphi\colon \mathbb{R} \to \mathbb{R}_+$ *is differentiable and* $\forall v \in \mathbb{R}$, $\varphi'(v) \neq 0$ *or* $\varphi'(-v) \neq 0$. *Then* $\ell_\varphi$ *can be expressed as a proper composite loss if and only if* $f\colon \mathbb{R} \ni v \mapsto -\frac{\varphi'(v)}{\varphi'(-v)}$ *is strictly monotonic continuous and* $\varphi$ *is monotonic.*

If $\varphi$ is convex or concave then $f$ defined above is monotonic. However not all binary margin losses are composite proper losses. One can even build a smooth margin loss which cannot be expressed as a proper composite loss. Consider $\varphi(x) = 1 - \frac{1}{\pi}\arctan(x-1)$. Then $f(v) = \frac{\varphi'(-v)}{\varphi'(-v)+\varphi'(v)} = \frac{x^2-2x+2}{2x^2+4}$ which is not invertible.

We now generalize the above results to the multiclass case.

**Proposition 17** *Suppose* $\ell$ *has two proper composite representations* $\ell = \lambda \circ \psi^{-1} = \mu \circ \phi^{-1}$ *where* $\lambda$ *and* $\mu$ *are proper losses and* $\psi$ *and* $\phi$ *are continuous invertible. Then* $\lambda = m$ *almost everywhere.*

*If* $\ell$ *is continuous and has a composite representation, then the proper loss (in the decomposition) is unique (* $\lambda = \mu$ *everywhere).*

*If* $\ell$ *is invertible and has a composite representation, then the representation is unique.*

Given a loss $\ell\colon \mathcal{V} \to \mathbb{R}^n_+$, we denote by $\mathscr{S}_\ell = \ell(\mathcal{V}) + [0,\infty)^n = \{\lambda\colon \exists v \in \mathcal{V},\ \forall i \in [n],\ \lambda_i \geq \ell_i(v)\}$ the *super-prediction set* of $\ell$ (confer e.g. [18]). We introduce a set of hyperplanes for $p \in \Delta^n$ and $\beta \in \mathbb{R}$, $h_p^\beta = \{x \in \mathbb{R}^n\colon x' \cdot p = \beta\}$. A hyperplane $h_p^\beta$ *supports* a set $\mathscr{A}$ at $x \in \mathscr{A}$ when $x \in h_p^\beta$ and for all $a \in \mathscr{A}$, $a' \cdot p \geq \beta$ or for all $a \in \mathscr{A}$, $a' \cdot p \leq \beta$. We say that $\mathscr{S}_\ell$ is *strictly convex in its inner part* when for all $p \in \Delta^n$, there exists an unique $x \in \ell(\mathcal{V})$ such that there exists a hyperplane $h_p^\beta$ supporting $\mathscr{S}_\ell$ at $x$. $\mathscr{S}_\ell$ is said to be *smooth* when for all $x \in \ell(\mathcal{V})$, there exists an unique hyperplane supporting $\mathscr{S}_\ell$ at $x$. If $\ell$ is invertible, we can express these two definitions in terms of $v \in \mathcal{V}$ rather than $x \in \ell(\mathcal{V})$. If $\ell\colon \mathcal{V} \to \mathbb{R}^n_+$ is strictly convex, then $\mathscr{S}_\ell$ will be strictly convex in its inner part.

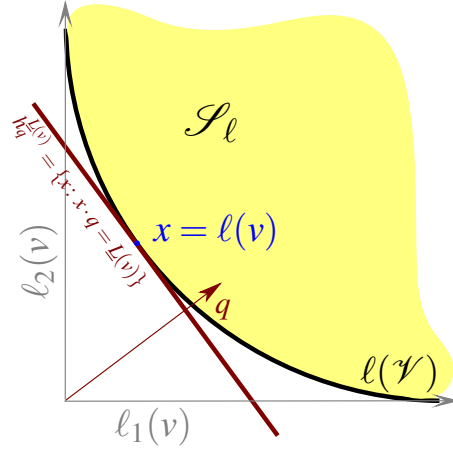

**Proposition 18** *Suppose* $\ell\colon \mathcal{V} \to \mathbb{R}^n_+$ *is a continuous invertible loss. Then* $\ell$ *has a strictly proper composite representation if and only if* $\mathscr{S}_\ell$ *is convex, smooth and strictly convex in its inner part.*

**Proposition 19** *Suppose* $\ell\colon \mathcal{V} \to \mathbb{R}^n_+$ *is a continuous loss. If* $\ell$ *has a proper composite representation, then* $\mathscr{S}_\ell$ *is convex and smooth. If* $\ell$ *is also invertible, then* $\mathscr{S}_\ell$ *is strictly convex in its inner part.*

## 6  Designing Proper Losses

We now build a family of conditional Bayes risks. Suppose we are given $\frac{n(n-1)}{2}$ concave functions $\{\underline{L}^{i_1,i_2}\colon \Delta^2 \to \mathbb{R}\}_{1 \leq i_1 < i_2 \leq n}$ on $\Delta^2$, and we want to build a concave function $\underline{L}$ on $\Delta^n$ which is equal to one of the given functions on each edge of the simplex ($\forall 1 \leq i_1 < i_2 \leq n$, $\underline{L}(0,.,0,p_{i_1},0,.,0,p_{i_2},0,.,0) = \underline{L}^{i_1,i_2}(p_{i_1},p_{i_2})$). This is equivalent to choosing a binary loss function, knowing that the observation is in the class $i_1$ or $i_2$. The result below gives one possible construction. (There exists an infinity of solutions — one can simply add any concave function equal to zero in each edge).

**Lemma 20** *Suppose we have a family of concave functions* $\{\underline{L}^{i_1,i_2}\colon \Delta^2 \to \mathbb{R}\}_{1 \leq i_1 < i_2 \leq n}$, *then*

$$\underline{L}\colon \Delta^n \ni p \mapsto \underline{L}(p_1,\ldots,p_n) = \sum_{1 \leq i_1 < i_2 \leq n} (p_{i_1} + p_{i_2})\underline{L}^{i_1,i_2}\left(\frac{p_{i_1}}{p_{i_1}+p_{i_2}}, \frac{p_{i_2}}{p_{i_1}+p_{i_2}}\right)$$

*is concave and* $\forall 1 \leq i_1 < i_2 \leq n$, $\underline{L}(0,.,0,p_{i_1},0,.,0,p_{i_2},0,.,0) = \underline{L}^{i_1,i_2}(p_{i_1},p_{i_2})$.

Using this family of Bayes risks, one can build a family of proper losses.

**Lemma 21** *Suppose we have a family of binary proper losses $\ell^{i_1,i_2} : \Delta^2 \to \mathbb{R}^2$. Then*

$$\ell : \Delta^n \ni p \mapsto \ell(p) = \left( \sum_{i=1}^{j-1} \ell_{-1}^{i,j} \left( \frac{p_i}{p_i + p_j} \right) + \sum_{i=j+1}^{n} \ell_{1}^{i,j} \left( \frac{p_j}{p_i + p_j} \right) \right)_{j=1}^{n} \in \mathbb{R}_+^n$$

*is a proper n-class loss such that*

$$\ell_i((0,.,.,0,p_{i_1},0,.,.,0,p_{i_2},0,.,.,0)) = \begin{cases} \ell_{1}^{i_1,i_2}(p_{i_1}) & i = i_1 \\ \ell_{-1}^{i_1,i_2}(p_{i_1}) & i = i_2 \\ 0 & otherwise \end{cases}.$$

Observe that it is much easier to work at first with the Bayes risk and then using the correspondence between Bayes risks and proper losses.

## 7 Integral Representations of Proper Losses

Unlike the natural generalisation of the results from proper binary to proper multiclass losses above, there is one result that does not carry over: the integral representation of proper losses [1]. In the binary case there exists a family of "extremal" loss functions (cost-weighted generalisations of the 0-1 loss) each parametrised by $c \in [0,1]$ and defined for all $\eta \in [0,1]$ by $\ell_{-1}^c(\eta) := c[\![\eta \geq c]\!]$ and $\ell_1^c := (1-c)[\![\eta < c]\!]$. As shown in [1, 3], given these extremal functions, any proper binary loss $\ell$ can be expressed as the weighted integral $\ell = \int_0^1 \ell^c w(c) \, dc + constant$ with $w(c) = -\underline{L}''(c)$. This representation is a special case of a representation from Choquet theory [19] which characterises when every point in some set can be expressed as a weighted combination of the "extremal points" of the set. Although there is such a representation when $n > 2$, the difficulty is that the set of extremal points is *much* larger and this rules out the existence of a nice small set of "primitive" proper losses when $n > 2$. The rest of this section makes this statement precise.

A *convex cone* $\mathcal{K}$ is a set of points closed under linear combinations of positive coefficients. That is, $\mathcal{K} = \alpha \mathcal{K} + \beta \mathcal{K}$ for any $\alpha, \beta \geq 0$. A point $f \in \mathcal{K}$ is *extremal* if $f = \frac{1}{2}(g + h)$ for $g, h \in \mathcal{K}$ implies $\exists \alpha \in \mathbb{R}_+$ such that $g = \alpha f$. That is, $f$ cannot be represented as a non-trivial combination of other points in $\mathcal{K}$. The set of extremal points for $\mathcal{K}$ will be denoted $\mathrm{ex}\,\mathcal{K}$. Suppose $U$ is a bounded closed convex set in $\mathbb{R}^d$, and $\mathcal{K}_b(U)$ is the set of convex functions on $U$ bounded by 1, then $\mathcal{K}_b(U)$ is compact with respect to the topology of uniform convergence. Theorem 2.2 of [20] shows that the extremal points of the convex cone $\mathcal{K}(U) = \{\alpha f + \beta g : f, g \in \mathcal{K}_b(U), \alpha, \beta \geq 0\}$ are dense (w.r.t. the topology of uniform convergence) in $\mathcal{K}(U)$ when $d > 1$. This means for any function $f \in \mathcal{K}(U)$ there is a sequence of functions $(g^i)_i$ such that for all $i$ $g^i \in \mathrm{ex}\,\mathcal{K}(U)$ and $\lim_{i \to \infty} \|f - g^i\|_\infty = 0$, where $\|f\|_\infty := \sup_{u \in U} |f(u)|$. We use this result to show that the set of extremal Bayes risks is dense in the set of Bayes risks when $n > 2$.

In order to simplify our analysis, we restrict attention to fair proper losses. A loss is *fair* if each partial loss is zero on its corresponding vertex of the simplex ($\ell_i(e_i) = 0, \forall i \in [n]$). A proper loss is fair if and only if its Bayes risk is zero at each vertex of the simplex (in this case the Bayes risk is also called fair). One does not lose generality by studying fair proper losses since any proper loss is a sum of a fair proper loss and a constant vector.

The set of fair proper losses defined on $\Delta^n$ form a closed convex cone, denoted $\mathscr{L}_n$. The set of concave functions which are zero on all the vertices of the simplex $\Delta^n$ is denoted $\mathscr{F}_n$ and is also a closed convex cone.

**Proposition 22** *Suppose $n > 2$. Then for any fair proper loss $\ell \in \mathscr{L}_n$ there exists a sequence $(\ell^i)_i$ of extremal fair proper losses ($\ell^i \in \mathrm{ex}\,\mathscr{L}_n$) which converges almost everywhere to $\ell$.*

The proof of Proposition 22 requires the following lemma which relies upon the correspondence between a proper loss and its Bayes risk (Proposition 11) and the fact that two continuous functions equal almost everywhere are equal everywhere.

**Lemma 23** *If $\ell \in \mathrm{ex}\,\mathscr{L}_n$ then its corresponding Bayes risk $\underline{L}$ is extremal in $\mathscr{F}_n$. Conversely, if $\underline{L} \in \mathrm{ex}\,\mathscr{F}_n$ then all the proper losses $\ell$ with Bayes risk equal to $\underline{L}$ are extremal in $\mathscr{L}_n$.*

We also need a correspondence between the uniform convergence of a sequence of Bayes risk functions and the convergence of their associated proper losses.

**Lemma 24** *Suppose $\underline{L}, \underline{L}^i \in \mathscr{F}_n$ for $i \in \mathbb{N}$ and suppose $\ell$ and $\ell^i$, $i \in \mathbb{N}$ are associated proper losses. Then $(\underline{L}^i)_i$ converges uniformly to $\underline{L}$ if and only if $(\ell^i)_i$ converges almost everywhere to $\ell$.*

Bronshtein [20] and Johansen [21] showed how to construct a set of extremal convex functions which is dense in $\mathscr{K}(U)$. With a trivial change of sign this leads to a family of extremal proper fair Bayes risks that is dense in the set of Bayes risks in the topology of uniform convergence. This means that it is not possible to have a small set of extremal ("primitive") losses from which one can construct any proper fair loss by linear combinations when $n > 2$.

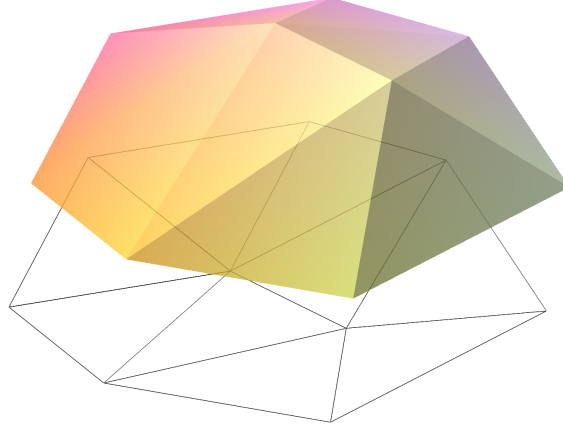

Figure 1: Complexity of extremal concave functions in two dimensions (corresponds to $n = 3$). Graph of an extremal concave function in two dimensions. Lines are where the slope changes. The pattern of these lines can be arbitrarily complex.

A convex *polytope* is a compact convex intersection of a finite set of half-spaces and is therefore the convex hull of its vertices. Let $\{a_i\}_i$ be a finite family of affine functions defined on $\Delta^n$. Now define the convex *polyhedral function* $f$ by $f(x) := \max_i a_i(x)$. The set $K := \{P_i = \{x \in \Delta^n : f(x) = a_i(x)\}\}$ is a covering of $\Delta^n$ by polytopes. Theorem 2.1 of [20] shows that for $f$, $P_i$ and $K$ so defined, $f$ is extremal if the following two conditions are satisfied: 1) for all polytopes $P_i$ in $K$ and for every face $F$ of $P_i$, $F \cap \Delta^n \neq \varnothing$ implies $F$ has a vertex in $\Delta^n$; 2) every vertex of $P_i$ in $\Delta^n$ belongs to $n$ distinct polytopes of $K$. The set of all such $f$ is dense in $\mathscr{K}(U)$.

Using this result it is straightforward to exhibit some sets of extremal fair Bayes risks $\{\underline{L}_c(p) : c \in \Delta^n\}$. Two examples are when $\underline{L}_c(p) = \sum_{i=1}^{n} \frac{p_i}{c_i} \prod_{j \neq i} [\![ \frac{p_i}{c_i} \leq \frac{p_j}{c_j} ]\!]$ or $\underline{L}_c(p) = \bigwedge_{i \in [n]} \frac{1 - p_i}{1 - c_i}$.

# 8 Conclusion

We considered loss functions for multiclass prediction problems and made four main contributions:

- We extended existing results for binary losses to multiclass prediction problems including several characterisations of proper losses and the relationship between properness and classification calibration;
- We related the notion of prediction calibration to classification calibration;
- We developed some new existence and uniqueness results for proper composite losses (which are new even in the binary case) which characterise when a loss has a proper composite representation in terms of the geometry of the associated superprediction set; and
- We showed that the attractive (simply parametrised) integral representation for binary proper losses can *not* be extended to the multiclass case.

Our results suggest that in order to design losses for multiclass prediction problems it is helpful to use the composite representation, and design the proper part via the Bayes risk as suggested for the binary case in [1]. The proper composite representation is used in [22].

## Acknowledgements

The work was performed whilst Elodie Vernet was visiting ANU and NICTA, and was supported by the Australian Research Council and NICTA, through backing Australia's ability.

# References

[1] Andreas Buja, Werner Stuetzle and Yi Shen. Loss functions for binary class probability estimation and classification: Structure and applications. Technical report, University of Pennsylvania, November 2005. `http://www-stat.wharton.upenn.edu/~buja/PAPERS/paper-proper-scoring.pdf`.

[2] Tilmann Gneiting and Adrian E. Raftery. Strictly proper scoring rules, prediction, and estimation. *Journal of the American Statistical Association*, 102(477):359-378, March 2007.

[3] Mark D. Reid and Robert C. Williamson. Information, divergence and risk for binary experiments. *Journal of Machine Learning Research*, 12:731-817, March 2011.

[4] Mark D. Reid and Robert C. Williamson. Composite binary losses. *Journal of Machine Learning Research*, 11:2387-2422, 2010.

[5] Peter L. Bartlett, Michael I. Jordan and Jon D. McAuliffe. Convexity, classification, and risk bounds. *Journal of the American Statistical Association*, 101(473):138-156, March 2006.

[6] Tong Zhang. Statistical analysis of some multi-category large margin classification methods. *Journal of Machine Learning Research*, 5:1225-1251, 2004.

[7] Simon I. Hill and Arnaud Doucet. A framework for kernel-based multi-category classification. *Journal of Artificial Intelligence Research*, 30:525-564, 2007.

[8] Ambuj Tewari and Peter L. Bartlett. On the consistency of multiclass classification methods. *Journal of Machine Learning Research*, 8:1007-1025, 2007.

[9] Yufeng Liu. Fisher consistency of multicategory support vector machines. *Proceedings of the Eleventh International Conference on Artificial Intelligence and Statistics*, side 289-296, 2007.

[10] Raúl Santos-Rodríguez, Alicia Guerrero-Curieses, Rocío Alaiz-Rodriguez and Jesús Cid-Sueiro. Cost-sensitive learning based on Bregman divergences. *Machine Learning*, 76:271-285, 2009. `http://dx.doi.org/10.1007/s10994-009-5132-8`.

[11] Hui Zou, Ji Zhu and Trevor Hastie. New multicategory boosting algorithms based on multicategory Fisher-consistent losses. *The Annals of Applied Statistics*, 2(4):1290-1306, 2008.

[12] Zhihua Zhang, Michael I. Jordan, Wu-Jun Li and Dit-Yan Yeung. Coherence functions for multicategory margin-based classification methods. *Proceedings of the Twelfth Conference on Artificial Intelligence and Statistics (AISTATS)*, 2009.

[13] Tobias Glasmachers. Universal consistency of multi-class support vector classication. *Advances in Neural Information Processing Systems (NIPS)*, 2010.

[14] Elodie Vernet, Robert C. Williamson and Mark D. Reid. Composite multiclass losses. (with proofs). To appear in NIPS 2011, October 2011. `http://users.cecs.anu.edu.au/~williams/papers/P188.pdf`.

[15] Jean-Baptiste Hiriart-Urruty and Claude Lemaréchal. *Fundamentals of Convex Analysis*. Springer, Berlin, 2001.

[16] Nicolas S. Lambert. Elicitation and evaluation of statistical forecasts. Technical report, Stanford University, March 2010. `http://www.stanford.edu/~nlambert/lambert_elicitation.pdf`.

[17] Jesús Cid-Sueiro and Aníbal R. Figueiras-Vidal. On the structure of strict sense Bayesian cost functions and its applications. *IEEE Transactions on Neural Networks*, 12(3):445-455, May 2001.

[18] Yuri Kalnishkan and Michael V. Vyugin. The weak aggregating algorithm and weak mixability. *Journal of Computer and System Sciences*, 74:1228-1244, 2008.

[19] Robert R. Phelps. *Lectures on Choquet's Theorem*, volume 1757 of *Lecture Notes in Mathematics*. Springer, 2nd edition, 2001.

[20] Efim Mikhailovich Bronshtein. Extremal convex functions. *Siberian Mathematical Journal*, 19:6-12, 1978.

[21] Søren Johansen. The extremal convex functions. *Mathematica Scandinavica*, 34:61-68, 1974.

[22] Tim van Erven, Mark D. Reid and Robert C. Williamson. Mixability is Bayes risk curvature relative to log loss. *Proceedings of the 24th Annual Conference on Learning Theory*, 2011. To appear. `http://users.cecs.anu.edu.au/~williams/papers/P186.pdf`.

[23] Rolf Schneider. *Convex Bodies: The Brunn-Minkowski Theory*. Cambridge University Press, 1993.

